# Incremental Algorithms
# for Hierarchical Classification[*]

**Nicolò Cesa-Bianchi**
Università di Milano
Milano, Italy

**Claudio Gentile**
Università dell'Insubria
Varese, Italy

**Andrea Tironi**  **Luca Zaniboni**
Università di Milano
Crema, Italy

## Abstract

We study the problem of hierarchical classification when labels corresponding to partial and/or multiple paths in the underlying taxonomy are allowed. We introduce a new hierarchical loss function, the H-loss, implementing the simple intuition that additional mistakes in the subtree of a mistaken class should not be charged for. Based on a probabilistic data model introduced in earlier work, we derive the Bayes-optimal classifier for the H-loss. We then empirically compare two incremental approximations of the Bayes-optimal classifier with a flat SVM classifier and with classifiers obtained by using hierarchical versions of the Perceptron and SVM algorithms. The experiments show that our simplest incremental approximation of the Bayes-optimal classifier performs, after just one training epoch, nearly as well as the hierarchical SVM classifier (which performs best). For the same incremental algorithm we also derive an H-loss bound showing, when data are generated by our probabilistic data model, exponentially fast convergence to the H-loss of the hierarchical classifier based on the true model parameters.

## 1 Introduction and basic definitions

We study the problem of classifying data in a given taxonomy of labels, where the taxonomy is specified as a tree forest. We assume that every data instance is labelled with a (possibly empty) set of class labels called *multilabel*, with the only requirement that multilabels including some node $i$ in the taxonomy must also include all ancestors of $i$. Thus, each multilabel corresponds to the union of one or more paths in the forest, where each path must start from a root but it can terminate on an internal node (rather than a leaf).

Learning algorithms for hierarchical classification have been investigated in, e.g., [8, 9, 10, 11, 12, 14, 15, 17, 20]. However, the scenario where labelling includes multiple and partial paths has received very little attention. The analysis in [5], which is mainly theoretical, shows in the multiple and partial path case a 0/1-loss bound for a hierarchical learning algorithm based on regularized least-squares estimates.

In this work we extend [5] in several ways. First, we introduce a new hierarchical loss function, the *H-loss*, which is better suited than the 0/1-loss to analyze hierarchical classification tasks, and we derive the corresponding Bayes-optimal classifier under the parametric data model introduced in [5]. Second, considering various loss functions, including the H-loss, we empirically compare the performance of the following three incremental kernel-based

---

[*]This work was supported in part by the PASCAL Network of Excellence under EC grant no. 506778. This publication only reflects the authors' views.

algorithms: 1) a hierarchical version of the classical Perceptron algorithm [16]; 2) an approximation to the Bayes-optimal classifier; 3) a simplified variant of this approximation. Finally, we show that, assuming data are indeed generated according to the parametric model mentioned before, the H-loss of the algorithm in 3) converges to the H-loss of the classifier based on the true model parameters. Our incremental algorithms are based on training linear-threshold classifiers in each node of the taxonomy. A similar approach has been studied in [8], though their model does not consider multiple-path classifications as we do.

Incremental algorithms are the main focus of this research, since we strongly believe that they are a key tool for coping with tasks where large quantities of data items are generated and the classification system needs to be frequently adjusted to keep up with new items. However, we found it useful to provide a reference point for our empirical results. Thus we have also included in our experiments the results achieved by nonincremental algorithms. In particular, we have chosen a flat and a hierarchical version of SVM [21, 7, 19], which are known to perform well on the textual datasets considered here.

We assume data elements are encoded as real vectors $\boldsymbol{x} \in \mathbb{R}^d$ which we call *instances*. A *multilabel* for an instance $\boldsymbol{x}$ is any subset of the set $\{1, \ldots, N\}$ of all labels/classes, including the empty set. We denote the multilabel associated with $\boldsymbol{x}$ by a vector $\boldsymbol{y} = (y_1, \ldots, y_N) \in \{0,1\}^N$, where $i$ belongs to the multilabel of $\boldsymbol{x}$ if and only if $y_i = 1$. A *taxonomy* $G$ is a forest whose trees are defined over the set of labels. A multilabel $\boldsymbol{y} \in \{0,1\}^N$ is said to *respect* a taxonomy $G$ if and only if $\boldsymbol{y}$ is the union of one or more paths in $G$, where each path starts from a root but need not terminate on a leaf. See Figure 1. We assume the data-generating mechanism produces *examples* $(\boldsymbol{x}, \boldsymbol{y})$ such that $\boldsymbol{y}$ respects some fixed underlying taxonomy $G$ with $N$ nodes. The set of roots in $G$ is denoted by $\mathrm{root}(G)$. We use $\mathrm{par}(i)$ to denote the unique parent of node $i$, $\mathrm{anc}(i)$ to denote the set of ancestors of $i$, and $\mathrm{sub}(i)$ to denote the set of nodes in the subtree rooted at $i$ (including $i$). Finally, given a predicate $\phi$ over a set $\Omega$, we will use $\{\phi\}$ to denote both the subset of $\Omega$ where $\phi$ is true and the indicator function of this subset.

## 2 The H-loss

Though several hierarchical losses have been proposed in the literature (e.g., in [11, 20]), no one has emerged as a standard yet. Since hierarchical losses are defined over multilabels, we start by considering two very simple functions measuring the discrepancy between multilabels $\widehat{\boldsymbol{y}} = (\widehat{y}_1, ..., \widehat{y}_N)$ and $\boldsymbol{y} = (y_1, ..., y_N)$: the 0/1-loss $\ell_{0/1}(\widehat{\boldsymbol{y}}, \boldsymbol{y}) = \{\exists i : \widehat{y}_i \neq y_i\}$ and the symmetric difference loss $\ell_\Delta(\widehat{\boldsymbol{y}}, \boldsymbol{y}) = \{\widehat{y}_1 \neq y_1\} + \ldots + \{\widehat{y}_N \neq y_N\}$.

There are several ways of making these losses depend on a given taxonomy $G$. In this work, we follow the intuition "if a mistake is made at node $i$, then further mistakes made in the subtree rooted at $i$ are unimportant". That is, we do not require the algorithm be able to make fine-grained distinctions on tasks when it is unable to make coarse-grained ones. For example, if an algorithm failed to label a document with the class SPORTS, then the algorithm should not be charged more loss because it also failed to label the same document with the subclass SOCCER and the sub-subclass CHAMPIONS LEAGUE. A function implementing this intuition is defined by

$$\ell_H(\widehat{\boldsymbol{y}}, \boldsymbol{y}) = \sum_{i=1}^N c_i \{\widehat{y}_i \neq y_i \wedge \widehat{y}_j = y_j, j \in \mathrm{anc}(i)\},$$

where $c_1, \ldots, c_N > 0$ are fixed *cost coefficients*. This loss, which we call H-loss, can also be described as follows: all paths in $G$ from a root down to a leaf are examined and, whenever we encounter a node $i$ such that $\widehat{y}_i \neq y_i$, we add $c_i$ to the loss, whereas all the loss contributions in the subtree rooted at $i$ are discarded. Note that if $c_1 = \ldots = c_N = 1$ then $\ell_{0/1} \leq \ell_H \leq \ell_\Delta$. Choices of $c_i$ depending on the structure of $G$ are proposed in Section 4. Given a multilabel $\widehat{\boldsymbol{y}} \in \{0,1\}^N$ define its *G-truncation* as the multilabel $\boldsymbol{y}' = (y'_1, ..., y'_N) \in \{0,1\}^N$ where, for each $i = 1, \ldots, N$, $y'_i = 1$ iff $\widehat{y}_i = 1$ and $\widehat{y}_j = 1$ for all $j \in \mathrm{anc}(i)$. Note that the *G*-truncation of any multilabel always respects *G*. A graphical

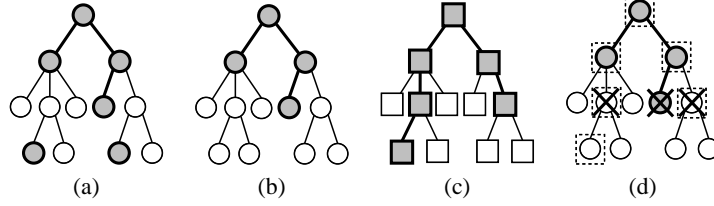

<div align="center">(a)        (b)        (c)        (d)</div>

Figure 1: A one-tree forest (repeated four times). Each node corresponds to a class in the taxonomy $G$, hence in this case $N = 12$. Gray nodes are included in the multilabel under consideration, white nodes are not. (a) A generic multilabel which *does not* respect $G$; (b) its $G$-truncation. (c) A second multilabel that respects $G$. (d) Superposition of multilabel (b) on multilabel (c): Only the checked nodes contribute to the H-loss between (b) and (c).

representation of the notions introduced so far is given in Figure 1. In the next lemma we show that whenever $\boldsymbol{y}$ respects $G$, then $\ell_H(\widehat{\boldsymbol{y}}, \boldsymbol{y})$ cannot be smaller than $\ell_H(\boldsymbol{y}', \boldsymbol{y})$. In other words, when the multilabel $\boldsymbol{y}$ to be predicted respects a taxonomy $G$ then there is no loss of generality in restricting to predictions which respect $G$.

**Lemma 1** *Let $G$ be a taxonomy, $\boldsymbol{y}, \widehat{\boldsymbol{y}} \in \{0,1\}^N$ be two multilabels such that $\boldsymbol{y}$ respects $G$, and $\boldsymbol{y}'$ be the $G$-truncation of $\widehat{\boldsymbol{y}}$. Then $\ell_H(\boldsymbol{y}', \boldsymbol{y}) \leq \ell_H(\widehat{\boldsymbol{y}}, \boldsymbol{y})$ .*

*Proof.* For each $i = 1, \ldots, N$ we show that $y_i' \neq y_i$ and $y_j' = y_j$ for all $j \in \text{anc}(i)$ implies $\widehat{y}_i \neq y_i$ and $\widehat{y}_j = y_j$ for all $j \in \text{anc}(i)$. Pick some $i$ and suppose $y_i' \neq y_i$ and $y_j' = y_j$ for all $j \in \text{anc}(i)$. Now suppose $y_j' = 0$ (and thus $y_j = 0$) for some $j \in \text{anc}(i)$. Then $y_i = 0$ since $\boldsymbol{y}$ respects $G$. But this implies $y_i' = 1$, contradicting the fact that the $G$-truncation $\boldsymbol{y}'$ respects $G$. Therefore, it must be the case that $y_j' = y_j = 1$ for all $j \in \text{anc}(i)$. Hence the $G$-truncation of $\widehat{\boldsymbol{y}}$ left each node $j \in \text{anc}(i)$ unchanged, implying $\widehat{y}_j = y_j$ for all $j \in \text{anc}(i)$. But, since the $G$-truncation of $\widehat{\boldsymbol{y}}$ does not change the value of a node $i$ whose ancestors $j$ are such that $\widehat{y}_j = 1$, this also implies $\widehat{y}_i = y_i'$. Therefore $\widehat{y}_i \neq y_i$ and the proof is concluded. $\square$

## 3  A probabilistic data model

Our learning algorithms are based on the following statistical model for the data, originally introduced in [5]. The model defines a probability distribution $f_G$ over the set of multilabels respecting a given taxonomy $G$ by associating with each node $i$ of $G$ a Bernoulli random variable $Y_i$ and defining

$$f_G(\boldsymbol{y} \mid \boldsymbol{x}) = \prod_{i=1}^N \mathbb{P}\left(Y_i = y_i \mid Y_{\text{par}(i)} = y_{\text{par}(i)}, \boldsymbol{X} = \boldsymbol{x}\right) .$$

To guarantee that $f_G(\boldsymbol{y} \mid \boldsymbol{x}) = 0$ whenever $\boldsymbol{y} \in \{0,1\}^N$ does not respect $G$, we set $\mathbb{P}\left(Y_i = 1 \mid Y_{\text{par}(i)} = 0, \boldsymbol{X} = \boldsymbol{x}\right) = 0$. Notice that this definition of $f_G$ makes the (rather simplistic) assumption that all $Y_k$ with the same parent node $i$ (i.e., the children of $i$) are independent when conditioned on $Y_i$ and $\boldsymbol{x}$. Through $f_G$ we specify an i.i.d. process $\{(\boldsymbol{X}_1, \boldsymbol{Y}_1), (\boldsymbol{X}_2, \boldsymbol{Y}_2), \ldots\}$, where, for $t = 1, 2, \ldots$, the multilabel $\boldsymbol{Y}_t$ is distributed according to $f_G(\cdot \mid \boldsymbol{X}_t)$ and $\boldsymbol{X}_t$ is distributed according to a fixed and unknown distribution $D$. Each example $(\boldsymbol{x}_t, \boldsymbol{y}_t)$ is thus a realization of the corresponding pair $(\boldsymbol{X}_t, \boldsymbol{Y}_t)$ of random variables. Our parametric model for $f_G$ is described as follows. First, we assume that the support of $D$ is the surface of the $d$-dimensional unit sphere (i.e., instances $\boldsymbol{x} \in \mathbb{R}^d$ are such that $||\boldsymbol{x}|| = 1$). With each node $i$ in the taxonomy, we associate a unit-norm weight vector $\boldsymbol{u}_i \in \mathbb{R}^d$. Then, we define the conditional probabilities for a nonroot node $i$ with parent $j$ by $\mathbb{P}\left(Y_i = 1 \mid Y_j = 1, \boldsymbol{X} = \boldsymbol{x}\right) = (1 + \boldsymbol{u}_i^\top \boldsymbol{x})/2$. If $i$ is a root node, the previous equation simplifies to $\mathbb{P}\left(Y_i = 1 \mid \boldsymbol{X} = \boldsymbol{x}\right) = (1 + \boldsymbol{u}_i^\top \boldsymbol{x})/2$.

### 3.1 The Bayes-optimal classifier for the H-loss

We now describe a classifier, called H-BAYES, that is the Bayes-optimal classifier for the H-loss. In other words, H-BAYES classifies any instance $\boldsymbol{x}$ with the multilabel $\widehat{\boldsymbol{y}} = \operatorname{argmin}_{\overline{\boldsymbol{y}} \in \{0,1\}} \mathbb{E}[\ell_H(\overline{\boldsymbol{y}}, \boldsymbol{Y}) \mid \boldsymbol{x}]$. Define $p_i(\boldsymbol{x}) = \mathbb{P}\left(Y_i = 1 \mid Y_{\text{par}(i)} = 1, \boldsymbol{X} = \boldsymbol{x}\right)$. When no ambiguity arises, we write $p_i$ instead of $p_i(\boldsymbol{x})$. Now, fix any unit-length instance $\boldsymbol{x}$ and let $\widehat{\boldsymbol{y}}$ be a multilabel that respects $G$. For each node $i$ in $G$, recursively define

$$\overline{H}_{i,\boldsymbol{x}}(\widehat{\boldsymbol{y}}) = c_i \left( p_i(1 - \widehat{y}_i) + (1 - p_i)\widehat{y}_i \right) + \sum_{k \in \text{child}(i)} \overline{H}_{k,\boldsymbol{x}}(\widehat{\boldsymbol{y}}) \ .$$

The classifier H-BAYES operates as follows. It starts by putting all nodes of $G$ in a set $S$; nodes are then removed from $S$ one by one. A node $i$ can be removed only if $i$ is a leaf or if all nodes $j$ in the subtree rooted at $i$ have been already removed. When $i$ is removed, its value $\widehat{y}_i$ is set to 1 if and only if

$$p_i \left( 2 - \sum_{k \in \text{child}(i)} \overline{H}_{k,\boldsymbol{x}}(\widehat{\boldsymbol{y}})/c_i \right) \geq 1 \ . \tag{1}$$

(Note that if $i$ is a leaf then (1) is equivalent to $\widehat{y}_i = \{p_i \geq 1/2\}$.) If $\widehat{y}_i$ is set to zero, then all nodes in the subtree rooted at $i$ are set to zero.

**Theorem 2** *For any taxonomy $G$ and all unit-length $\boldsymbol{x} \in \mathbb{R}^d$, the multilabel generated by H-BAYES is the Bayes-optimal classification of $\boldsymbol{x}$ for the H-loss.*

*Proof sketch.* Let $\widehat{\boldsymbol{y}}$ be the multilabel assigned by H-BAYES and $\boldsymbol{y}^*$ be any multilabel minimizing the expected H-loss. Introducing the short-hand $\mathbb{E}_{\boldsymbol{x}}[\cdot] = \mathbb{E}[\cdot \mid \boldsymbol{x}]$, we can write

$$\mathbb{E}_{\boldsymbol{x}} \, \ell_H(\widehat{\boldsymbol{y}}, \boldsymbol{Y}) = \sum_{i=1}^{N} c_i \left( p_i(1 - \widehat{y}_i) + (1 - p_i)\widehat{y}_i \right) \prod_{j \in \text{anc}(i)} p_j \{\widehat{y}_j = 1\} \ .$$

Note that we can recursively decompose the expected H-loss as

$$\mathbb{E}_{\boldsymbol{x}} \, \ell_H(\widehat{\boldsymbol{y}}, \boldsymbol{Y}) = \sum_{i \in \text{root}(G)} \mathbb{E}_{\boldsymbol{x}} \, H_i(\widehat{\boldsymbol{y}}, \boldsymbol{Y}),$$

where
$$\mathbb{E}_{\boldsymbol{x}} H_i(\widehat{\boldsymbol{y}}, \boldsymbol{Y}) = c_i \left( p_i(1 - \widehat{y}_i) + (1 - p_i)\widehat{y}_i \right) \prod_{j \in \text{anc}(i)} p_j \{\widehat{y}_j = 1\} + \sum_{k \in \text{child}(i)} \mathbb{E}_{\boldsymbol{x}} H_k(\widehat{\boldsymbol{y}}, \boldsymbol{Y}) \ . \tag{2}$$

Pick a node $i$. If $i$ is a leaf, then the sum in the RHS of (2) disappears and $y_i^* = \{p_i \geq 1/2\}$, which is also the minimizer of $\overline{H}_{i,\boldsymbol{x}}(\widehat{\boldsymbol{y}}) = c_i \left( p_i(1 - \widehat{y}_i) + (1 - p_i)\widehat{y}_i \right)$, implying $\widehat{y}_i = y_i^*$. Now let $i$ be an internal node and inductively assume $\widehat{y}_j = y_j^*$ for all $j \in \text{sub}(i)$. Notice that the factors $\prod_{j \in \text{anc}(i)} p_j \{\widehat{y}_j = 1\}$ occur in both terms in the RHS of (2). Hence $y_i^*$ does not depend on these factors and we can equivalently minimize
$$c_i \left( p_i(1 - \widehat{y}_i) + (1 - p_i)\widehat{y}_i \right) + p_i \{\widehat{y}_i = 1\} \sum_{k \in \text{child}(i)} \overline{H}_{k,\boldsymbol{x}}(\widehat{\boldsymbol{y}}), \tag{3}$$

where we noted that, for each $k \in \text{child}(i)$,

$$\mathbb{E}_{\boldsymbol{x}} H_k(\widehat{\boldsymbol{y}}, \boldsymbol{Y}) = \left( \prod_{j \in \text{anc}(i)} p_j \{\widehat{y}_j = 1\} \right) p_i \{\widehat{y}_i = 1\} \overline{H}_{k,\boldsymbol{x}}(\widehat{\boldsymbol{y}}) \ .$$

Now observe that $y_i^*$ minimizing (3) is equivalent to the assignment produced by H-BAYES. To conclude the proof, note that whenever $y_i^* = 0$, Lemma 1 requires that $y_j^* = 0$ for all nodes $j \in \text{sub}(i)$, which is exactly what H-BAYES does. $\qquad\square$

## 4 The algorithms

We consider three incremental algorithms. Each one of these algorithms learns a hierarchical classifier by training a decision function $g_i : \mathbb{R}^d \to \{0, 1\}$ at each node $i = 1, \ldots, N$. For a given set $g_1, \ldots, g_N$ of decision functions, the hierarchical classifier generated by these algorithms classifies an instance $\boldsymbol{x}$ through a multilabel $\widehat{\boldsymbol{y}} = (\widehat{y}_1, ..., \widehat{y}_N)$ defined as follows:

$$\widehat{y}_i = \begin{cases} g_i(\boldsymbol{x}) & \text{if } i \in \text{root}(G) \text{ or } \widehat{y}_j = 1 \text{ for all } j \in \text{anc}(i) \\ 0 & \text{otherwise.} \end{cases} \tag{4}$$

Note that $\widehat{\boldsymbol{y}}$ computed this way respects $G$. The classifiers (4) are trained incrementally. Let $g_{i,t}$ be the decision function at node $i$ after training on the first $t-1$ examples. When the next training example $(\boldsymbol{x}_t, \boldsymbol{y}_t)$ is available, the algorithms compute the multilabel $\widehat{\boldsymbol{y}}_t$ using classifier (4) based on $g_{1,t}(\boldsymbol{x}_t), \ldots, g_{N,t}(\boldsymbol{x}_t)$. Then, the algorithms consider for an update only those decision functions sitting at nodes $i$ satisfying either $i \in \text{root}(G)$ or $y_{\text{par}(i),t} = 1$. We call such nodes *eligible at time* $t$. The decision functions of all other nodes are left unchanged. The first algorithm we consider is a simple hierarchical version of the Perceptron algorithm [16], which we call H-PERC. The decision functions at time $t$ are defined by $g_{i,t}(\boldsymbol{x}_t) = \{\boldsymbol{w}_{i,t}^\top \boldsymbol{x}_t \geq 0\}$. In the update phase, the Perceptron rule $\boldsymbol{w}_{i,t+1} = \boldsymbol{w}_{i,t} + y_{i,t}\boldsymbol{x}_t$ is applied to every node $i$ eligible at time $t$ and such that $\widehat{y}_{i,t} \neq y_{i,t}$. The second algorithm, called APPROX-H-BAYES, approximates the H-BAYES classifier of Section 3.1 by replacing the unknown quantities $p_i(\boldsymbol{x}_t)$ with estimates $(1 + \boldsymbol{w}_{i,t}^\top \boldsymbol{x}_t)/2$. The weights $\boldsymbol{w}_{i,t}$ are regularized least-squares estimates defined by

$$\boldsymbol{w}_{i,t} = (I + S_{i,t-1} S_{i,t-1}^\top + \boldsymbol{x}_t \boldsymbol{x}_t^\top)^{-1} S_{i,t-1} \boldsymbol{y}_{t-1}^{(i)} . \tag{5}$$

The columns of the matrix $S_{i,t-1}$ are all past instances $\boldsymbol{x}_s$ that have been stored at node $i$; the $s$-th component of vector $\boldsymbol{y}_{t-1}^{(i)}$ is the $i$-th component $y_{i,s}$ of the multilabel $\boldsymbol{y}_s$ associated with instance $\boldsymbol{x}_s$. In the update phase, an instance $\boldsymbol{x}_t$ is stored at node $i$, causing an update of $\boldsymbol{w}_{i,t}$, whenever $i$ is eligible at time $t$ and $|\boldsymbol{w}_{i,t}^\top \boldsymbol{x}_t| \leq \sqrt{(5 \ln t)/N_{i,t}}$, where $N_{i,t}$ is the number of instances stored at node $i$ up to time $t-1$. The corresponding decision functions $g_{i,t}$ are of the form $g_{i,t}(\boldsymbol{x}_t) = \{\boldsymbol{w}_{i,t}^\top \boldsymbol{x}_t \geq \tau_{i,t}\}$, where the threshold $\tau_{i,t} \geq 0$ at node $i$ depends on the margin values $\boldsymbol{w}_{j,t}^\top \boldsymbol{x}_t$ achieved by nodes $j \in \text{sub}(i)$ — recall (1). Note that $g_{i,t}$ is not a linear-threshold function, as $\boldsymbol{x}_t$ appears in the definition of $\boldsymbol{w}_{i,t}$. The margin threshold $\sqrt{(5 \ln t)/N_{i,t}}$, controlling the update of node $i$ at time $t$, reduces the space requirements of the classifier by keeping matrices $S_{i,t}$ suitably small. This threshold is motivated by the work [4] on selective sampling.

The third algorithm, which we call H-RLS (Hierarchical Regularized Least Squares), is a simplified variant of APPROX-H-BAYES in which the thresholds $\tau_{i,t}$ are set to zero. That is, we have $g_{i,t}(\boldsymbol{x}_t) = \{\boldsymbol{w}_{i,t}^\top \boldsymbol{x}_t \geq 0\}$ where the weights $\boldsymbol{w}_{i,t}$ are defined as in (5) and updated as in the APPROX-H-BAYES algorithm. Details on how to run APPROX-H-BAYES and H-RLS in dual variables and perform an update at node $i$ in time $O(N_{i,t}^2)$ are found in [3] (where a mistake-driven version of H-RLS is analyzed).

## 5 Experimental results

The empirical evaluation of the algorithms was carried out on two well-known datasets of free-text documents. The first dataset consists of the first (in chronological order) 100,000 newswire stories from the Reuters Corpus Volume 1, RCV1 [2]. The associated taxonomy of labels, which are the topics of the documents, has 101 nodes organized in a forest of 4 trees. The forest is shallow: the longest path has length 3 and the the distribution of nodes, sorted by increasing path length, is $\{0.04, 0.53, 0.42, 0.01\}$. For this dataset, we used the bag-of-words vectorization performed by Xerox Research Center Europe within the EC project KerMIT (see [4] for details on preprocessing). The 100,000 documents were divided into 5 equally sized groups of chronologically consecutive documents. We then used each adjacent pair of groups as training and test set in an experiment (here the fifth and first group are considered adjacent), and then averaged the test set performance over the 5 experiments.

The second dataset is a specific subtree of the OHSUMED corpus of medical abstracts [1]: the subtree rooted in "Quality of Health Care" (MeSH code N05.715). After removing overlapping classes (OHSUMED is not quite a tree but a DAG), we ended up with 94

Table 1: Experimental results on two hierarchical text classification tasks under various loss functions. We report average test errors along with standard deviations (in parenthesis). In bold are the best performance figures among the incremental algorithms.

| RCV1 | 0/1-loss | unif. H-loss | norm. H-loss | $\Delta$-loss |
|---|---|---|---|---|
| PERC | $0.702(\pm0.045)$ | $1.196(\pm0.127)$ | $0.100(\pm0.029)$ | $1.695(\pm0.182)$ |
| H-PERC | $0.655(\pm0.040)$ | $1.224(\pm0.114)$ | $0.099(\pm0.028)$ | $1.861(\pm0.172)$ |
| H-RLS | $\mathbf{0.456(\pm0.010)}$ | $\mathbf{0.743(\pm0.026)}$ | $\mathbf{0.057(\pm0.001)}$ | $\mathbf{1.086(\pm0.036)}$ |
| AH-BAY | $0.550(\pm0.010)$ | $0.815(\pm0.028)$ | $0.090(\pm0.001)$ | $1.465(\pm0.040)$ |
| SVM | $0.482(\pm0.009)$ | $0.790(\pm0.023)$ | $0.057(\pm0.001)$ | $1.173(\pm0.051)$ |
| H-SVM | $0.440(\pm0.008)$ | $0.712(\pm0.021)$ | $0.055(\pm0.001)$ | $1.050(\pm0.027)$ |

| OHSU. | 0/1-loss | unif. H-loss | norm. H-loss | $\Delta$-loss |
|---|---|---|---|---|
| PERC | $0.899(\pm0.024)$ | $1.938(\pm0.219)$ | $0.058(\pm0.005)$ | $2.639(\pm0.226)$ |
| H-PERC | $0.846(\pm0.024)$ | $1.560(\pm0.155)$ | $0.057(\pm0.005)$ | $2.528(\pm0.251)$ |
| H-RLS | $\mathbf{0.769(\pm0.004)}$ | $1.200(\pm0.007)$ | $\mathbf{0.045(\pm0.000)}$ | $\mathbf{1.957(\pm0.011)}$ |
| AH-BAY | $0.819(\pm0.004)$ | $\mathbf{1.197(\pm0.006)}$ | $0.047(\pm0.000)$ | $2.029(\pm0.009)$ |
| SVM | $0.784(\pm0.003)$ | $1.206(\pm0.003)$ | $0.044(\pm0.000)$ | $1.872(\pm0.005)$ |
| H-SVM | $0.759(\pm0.002)$ | $1.170(\pm0.005)$ | $0.044(\pm0.000)$ | $1.910(\pm0.007)$ |

classes and 55,503 documents. We made this choice based only on the structure of the subtree: the longest path has length 4, the distribution of nodes sorted by increasing path length is $\{0.26, 0.37, 0.22, 0.12, 0.03\}$, and there are a significant number of partial and multiple path multilabels. The vectorization of the subtree was carried out as follows: after tokenization, we removed all stopwords and also those words that did not occur at least 3 times in the corpus. Then, we vectorized the documents using the Bow library [13] with a $\log(1+\text{TF})\log(\text{IDF})$ encoding. We ran 5 experiments by randomly splitting the corpus in a training set of 40,000 documents and a test set of 15,503 documents. Test set performances are averages over these 5 experiments. In the training set we kept more documents than in the RCV1 splits since the OHSUMED corpus turned out to be a harder classification problem than RCV1. In both datasets instances have been normalized to unit length. We tested the hierarchical Perceptron algorithm (H-PERC), the hierarchical regularized least-squares algorithm (H-RLS), and the approximated Bayes-optimal algorithm (APPROX-H-BAYES), all described in Section 4. The results are summarized in Table 1. APPROX-H-BAYES (AH-BAY in Table 1) was trained using cost coefficients $c_i$ chosen as follows: if $i \in \text{root}(G)$ then $c_i = |\text{root}(G)|^{-1}$. Otherwise, $c_i = c_j/|\text{child}(j)|$, where $j$ is the parent of $i$. Note that this choice of coefficients amounts to splitting a unit cost equally among the roots and then splitting recursively each node's cost equally among its children. Since, in this case, $0 \le \ell_H \le 1$, we call the resulting loss normalized H-loss. We also tested a hierarchical version of SVM (denoted by H-SVM in Table 1) in which each node is an SVM classifier trained using a batch version of our hierarchical learning protocol. More precisely, each node $i$ was trained only on those examples $(\boldsymbol{x}_t, \boldsymbol{y}_t)$ such that $y_{\text{par}(i),t} = 1$ (note that, as no conditions are imposed on $y_{i,t}$, node $i$ is actually trained on both positive and negative examples). The resulting set of linear-threshold functions was then evaluated on the test set using the hierarchical classification scheme (4). We tried both the $C$ and $\nu$ parametrizations [18] for SVM and found the setting $C = 1$ to work best for our data.[1] We finally tested the "flat" variants of Perceptron and SVM, denoted by PERC and SVM. In these variants, each node is trained and evaluated independently of the others, disregarding all taxonomical information. All SVM experiments were carried out using the libSVM implementation [6]. All the tested algorithms used a linear kernel.

As far as loss functions are concerned, we considered the 0/1-loss, the H-loss with cost coefficients set to 1 (denoted by uniform H-loss), the normalized H-loss, and the symmetric difference loss (denoted by $\Delta$-loss). Note that H-SVM performs best, but our incremental algorithms were trained for a single epoch on the training set. The good performance of SVM (the flat variant of H-SVM) is surprising. However, with a single epoch of training H-RLS does not perform worse than SVM (except on OHSUMED under the normalized H-loss) and comes reasonably close to H-SVM. On the other hand, the performance of APPROX-H-BAYES is disappointing: on OHSUMED it is the best algorithm only for the uniform H-loss, though it was trained using the normalized H-loss; on RCV1 it never outperforms H-RLS, though it always does better than PERC and H-PERC. A possible explanation for this behavior is that APPROX-H-BAYES is very sensitive to errors in the estimates of $p_i(\boldsymbol{x})$ (recall Section 3.1). Indeed, the least-squares estimates (5), which we used to approximate H-BAYES, seem to work better in practice on simpler (and possibly more robust) algorithms, such as H-RLS. The lower values of normalized H-loss on OHSUMED (a harder corpus than RCV1) can be explained because a quarter of the 94 nodes in the OHSUMED taxonomy are roots, and thus each top-level mistake is only charged about $4/94$. As a final remark, we observe that the normalized H-loss gave too small a range of values to afford fine comparisons among the best performing algorithms.

## 6 Regret bounds for the H-loss

In this section we prove a theoretical bound on the H-loss of a slight variant of the algorithm H-RLS tested in Section 5. More precisely, we assume data are generated according to the probabilistic model introduced in Section 3 with unknown instance distribution $D$ and unknown coefficients $\boldsymbol{u}_1, \ldots, \boldsymbol{u}_N$. We define the regret of a classifier assigning label $\widehat{\boldsymbol{y}}$ to instance $\boldsymbol{X}$ as $\mathbb{E}\,\ell_H(\widehat{\boldsymbol{y}}, \boldsymbol{Y}_t) - \mathbb{E}\,\ell_H(\overline{\boldsymbol{y}}, \boldsymbol{Y})$, where the expected value is with respect the random draw of $(\boldsymbol{X}, \boldsymbol{Y})$ and $\overline{\boldsymbol{y}}$ is the multilabel assigned by classifier (4) when the decision functions $g_i$ are zero-threshold functions of the form $g_i(\boldsymbol{x}) = \{\boldsymbol{u}_i^\top \boldsymbol{x} \geq 0\}$. The theorem below shows that the regret of the classifier learned by a variant of H-RLS after $t$ training examples, with $t$ large enough, is exponentially small in $t$. In other words, H-RLS learns to classify as well as the algorithm that is given the true parameters $\boldsymbol{u}_1, \ldots, \boldsymbol{u}_N$ of the underlying data-generating process. We have been able to prove the theorem only for the variant of H-RLS storing all instances at each node. That is, every eligible node at time $t$ is updated, irrespective of whether $|\boldsymbol{w}_{i,t}^\top \boldsymbol{x}_t| \leq \sqrt{(5 \ln t)/N_{i,t}}$.

Given the i.i.d. data-generating process $(\boldsymbol{X}_1, \boldsymbol{Y}_1), (\boldsymbol{X}_2, \boldsymbol{Y}_2), \ldots$, for each node $k$ we define the derived process $\boldsymbol{X}_{k_1}, \boldsymbol{X}_{k_2}, \ldots$ including all and only the instances $\boldsymbol{X}_s$ of the original process that satisfy $Y_{\mathrm{par}(k),s} = 1$. We call this derived process the *process at node $k$*. Note that, for each $k$, the process at node $k$ is an i.i.d. process. However, its distribution might depend on $k$. The spectrum of the process at node $k$ is the set of eigenvalues of the correlation matrix with entries $\mathbb{E}[X_{k_1,i} X_{k_1,j}]$ for $i, j = 1, \ldots, d$. We have the following theorem, whose proof is omitted due to space limitations.

**Theorem 3** *Let $G$ be a taxonomy with $N$ nodes and let $f_G$ be a joint density for $G$ parametrized by $N$ unit-norm vectors $\boldsymbol{u}_1, \ldots, \boldsymbol{u}_N \in \mathbb{R}^d$. Assume the instance distribution is such that there exist $\gamma_1, \ldots, \gamma_N > 0$ satisfying $\mathbb{P}\left(|\boldsymbol{u}_i^\top \boldsymbol{X}_t| \geq \gamma_i\right) = 1$ for $i = 1, \ldots, N$. Then, for all $t > \max\left\{\max_{i=1,\ldots,N} \frac{16}{\mu_i \lambda_i \gamma_i}, \max_{i=1,\ldots,N} \frac{192d}{\mu_i \lambda_i^2}\right\}$ the regret $\mathbb{E}\,\ell_H(\widehat{\boldsymbol{y}}_t, \boldsymbol{Y}_t) - \mathbb{E}\,\ell_H(\overline{\boldsymbol{y}}_t, \boldsymbol{Y}_t)$ of the modified H-RLS algorithm is at most*

$$\sum_{i=1}^N \mu_i \left[ t\, e^{-\kappa_1 \gamma_i^2 \lambda_i t \mu_i} + t^2\, e^{-\kappa_2 \lambda_i^2 t \mu_i} \right] \left( \sum_{j \in sub(i)} c_j \right),$$

*where $\kappa_1, \kappa_2$ are constants, $\mu_i = \mathbb{E}\left[\prod_{j \in anc(i)} \left((1 + \boldsymbol{u}_j^\top \boldsymbol{X})/2\right)\right]$ and $\lambda_i$ is the smallest eigenvalue in the spectrum of the process at node $i$.*

# 7 Conclusions and open problems

In this work we have studied the problem of hierarchical classification of data instances in the presence of partial and multiple path labellings. We have introduced a new hierarchical loss function, the H-loss, derived the corresponding Bayes-optimal classifier, and empirically compared an incremental approximation to this classifier with some other incremental and nonincremental algorithms. Finally, we have derived a theoretical guarantee on the H-loss of a simplified variant of the approximated Bayes-optimal algorithm.

Our investigation leaves several open issues. The current approximation to the Bayes-optimal classifier is not satisfying, and this could be due to a bad choice of the model, of the estimators, of the datasets, or of a combination of them. Also, the normalized H-loss is not fully satisfying, since the resulting values are often too small. From the theoretical viewpoint, we would like to analyze the regret of our algorithms with respect to the Bayes-optimal classifier, rather than with respect to a classifier that makes a suboptimal use of the true model parameters.

## Footnotes

[1]It should be emphasized that this tuning of $C$ was actually chosen in hindsight, with no cross-validation.

# References

[1] The OHSUMED test collection. URL: medir.ohsu.edu/pub/ohsumed/.

[2] Reuters corpus volume 1. URL: about.reuters.com/researchandstandards/corpus/.

[3] N. Cesa-Bianchi, A. Conconi, and C. Gentile. A second-order Perceptron algorithm. In *Proc. 15th COLT*, pages 121–137. Springer, 2002.

[4] N. Cesa-Bianchi, A. Conconi, and C. Gentile. Learning probabilistic linear-threshold classifiers via selective sampling. In *Proc. 16th COLT*, pages 373–386. Springer, 2003.

[5] N. Cesa-Bianchi, A. Conconi, and C. Gentile. Regret bounds for hierarchical classification with linear-threshold functions. In *Proc. 17th COLT*. Springer, 2004. To appear.

[6] C.-C. Chang and C.-J. Lin. Libsvm — a library for support vector machines. URL: www.csie.ntu.edu.tw/~cjlin/libsvm/.

[7] N. Cristianini and J. Shawe-Taylor. *An Introduction to Support Vector Machines*. Cambridge University Press, 2001.

[8] O. Dekel, J. Keshet, and Y. Singer. Large margin hierarchical classification. In *Proc. 21st ICML*. Omnipress, 2004.

[9] S.T. Dumais and H. Chen. Hierarchical classification of web content. In *Proc. 23rd ACM Int. Conf. RDIR*, pages 256–263. ACM Press, 2000.

[10] M. Granitzer. *Hierarchical Text Classification using Methods from Machine Learning*. PhD thesis, Graz University of Technology, 2003.

[11] T. Hofmann, L. Cai, and M. Ciaramita. Learning with taxonomies: Classifying documents and words. In *NIPS Workshop on Syntax, Semantics, and Statistics*, 2003.

[12] D. Koller and M. Sahami. Hierarchically classifying documents using very few words. In *Proc. 14th ICML*, Morgan Kaufmann, 1997.

[13] A. McCallum. Bow: A toolkit for statistical language modeling, text retrieval, classification and clustering. URL: www-2.cs.cmu.edu/~mccallum/bow/.

[14] A.K. McCallum, R. Rosenfeld, T.M. Mitchell, and A.Y. Ng. Improving text classification by shrinkage in a hierarchy of classes. In *Proc. 15th ICML*. Morgan Kaufmann, 1998.

[15] D. Mladenic. Turning yahoo into an automatic web-page classifier. In *Proceedings of the 13th European Conference on Artificial Intelligence*, pages 473–474, 1998.

[16] F. Rosenblatt. The Perceptron: A probabilistic model for information storage and organization in the brain. *Psychol. Review*, 65:386–408, 1958.

[17] M.E. Ruiz and P. Srinivasan. Hierarchical text categorization using neural networks. *Information Retrieval*, 5(1):87–118, 2002.

[18] B. Schölkopf, A. J. Smola, R. C. Williamson, and P. L. Bartlett. New support vector algorithms. *Neural Computation*, 12:1207–1245, 2000.

[19] B. Schölkopf and A. Smola. *Learning with kernels*. MIT Press, 2002.

[20] A. Sun and E.-P. Lim. Hierarchical text classification and evaluation. In *Proc. 2001 Int. Conf. Data Mining*, pages 521–528. IEEE Press, 2001.

[21] V.N. Vapnik. *Statistical Learning Theory*. Wiley, 1998.
